# Silicon Growth Cones Map Silicon Retina

**Brian Taba and Kwabena Boahen**[*]
Department of Bioengineering
University of Pennsylvania
Philadelphia, PA 19104
{btaba,boahen}@seas.upenn.edu

## Abstract

We demonstrate the first fully hardware implementation of retinotopic self-organization, from photon transduction to neural map formation. A silicon retina transduces patterned illumination into correlated spike trains that drive a population of silicon growth cones to automatically wire a topographic mapping by migrating toward sources of a diffusible guidance cue that is released by postsynaptic spikes. We varied the pattern of illumination to steer growth cones projected by different retinal ganglion cell types to self-organize segregated or coordinated retinotopic maps.

## 1   Introduction

Engineers have long admired the brain's ability to effortlessly adapt to novel situations without instruction, and sought to endow digital computers with a similar capacity for unsupervised self-organization. One prominent example is Kohonen's self-organizing map [1], which achieved popularity by distilling neurophysiological insights into a simple set of mathematical equations. Although these algorithms are readily simulated in software, previous hardware implementations have required high precision components that are expensive in chip area (e.g. [2, 3]). By contrast, neurobiological systems can self-organize components possessing remarkably heterogeneous properties. To pursue this biological robustness against component mismatch, we designed circuits that mimic neurophysiological function down to the subcellular level. In this paper, we demonstrate topographic refinement of connections between a silicon retina and the first neuromorphic self-organizing map chip, previously reported in [5], which is based on axon migration in the developing brain.

During development, neurons wire themselves into their mature circuits by extending axonal and dendritic precursors called *neurites*. Each neurite is tipped by a motile sensory structure called a *growth cone* that guides the elongating neurite based on local chemical cues. Growth cones move by continually sprouting and retracting finger-like extensions called *filopodia* whose dynamics can be biased by diffusible ligands in an activity-dependent manner [4]. Based on these observations, we designed and fabricated the Neurotrope1 chip to implement a population of silicon growth cones [5]. We interfaced Neu-

---

[*]www.neuroengineering.upenn.edu/boahen

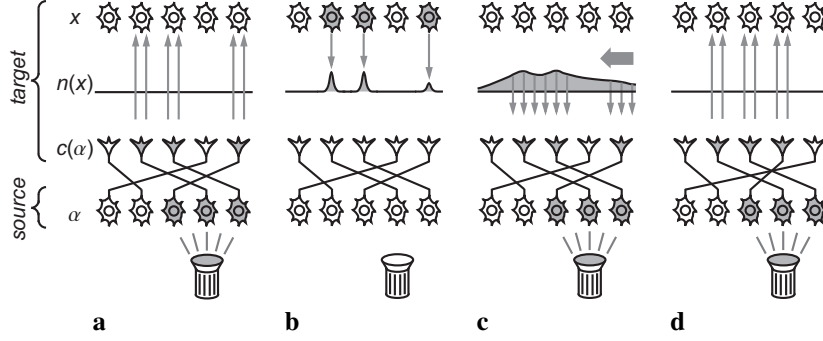

Figure 1: Neurotropic axon guidance. **a.** Active source cells (grey) relay spikes down their axons to their growth cones, which excite nearby target cells. **b.** Active target cell bodies secrete neurotropin. **c.** Neurotropin spreads laterally, establishing a spatial concentration gradient that is sampled by active growth cones. **d.** Active growth cones climb the local neurotropin gradient, translating temporal activity coincidence into spatial position coincidence. Growth cones move by displacing other growth cones.

rotrope1 directly to a spiking silicon retina to illustrate its applicability to larger neuromorphic systems.

This paper is organized as follows. In Section 2, we present an algorithm for axon migration under the guidance of a diffusible chemical whose release and uptake is gated by activity. In Section 3, we describe our hardware implementation of this algorithm. In Section 4, we examine the Neurotrope1 system's performance on a topographic refinement task when driven by spike trains generated by a silicon retina in response to several types of illumination stimuli.

## 2   Neurotropic axon guidance

We model the self-organization of connections between two layers of neurons (Fig. 1). Cells in the *source layer* innervate cells in the *target layer* with excitatory axons that are tipped by motile growth cones. Growth cones tow their axons within the target layer as directed by a diffusible guidance factor called *neurotropin* that they bind from the local extracellular environment. Neurotropin is released by postsynaptically active target cell bodies and bound by presynaptically active growth cones, so the retrograde transfer of neurotropin from a target cell to a source cell measures the temporal coincidence of their spike activities. Growth cones move to maximize their neurotropic uptake, a Hebbian-like learning rule that causes cells that fire at the same time to wire to the same place. To prevent the population of growth cones from attempting to trivially maximize their uptake by all exciting the same target cell, we impose a synaptic density constraint that requires a migrating growth cone to displace any other growth cone occupying its path.

To state the model more formally, source cell bodies occupy nodes of a regular two-dimensional (2D) lattice embedded in the source layer, while growth cones and target cell bodies occupy nodes on separate 2D lattices that are interleaved in the target layer. We index nodes by their positions in their respective layers, using Greek letters for source layer positions (e.g., $\alpha \in \mathbb{Z}^2$) and Roman letters for target layer positions (e.g., $x, c \in \mathbb{Z}^2$).

Each source cell $\alpha$ fires spikes at a rate $a_{\text{SC}}(\alpha)$ and conveys this presynaptic activity down an axon that elaborates an excitatory arbor in the target layer centered on $c(\alpha)$. In principle, every branch of this arbor is tipped by its own motile growth cone, but to facilitate efficient

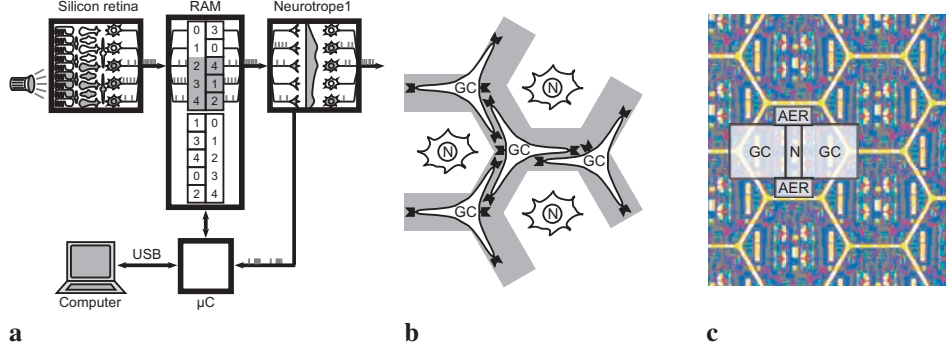

Figure 2: **a.** Neurotrope1 system. Spike communication is by address-events (AER). **b.** Neurotrope1 cell mosaic. The extracellular medium (grey) is laid out as a monolithic honeycomb lattice. Growth cones (GC) occupy nodes of this lattice and extend filopodia to the adjacent nodes. Neurotropin receptors (black) are located at the tip of each filopodium and at the growth cone body. Target cells (N) occupy nodes of an interleaved triangular lattice. **c.** Detail of chip layout.

hardware implementation, we abstract the collection of branch growth cones into a single central growth cone that tows the arbor's trunk around the target layer, dragging the rest of the arbor with it. The arbor overlaps nearby target cells with a branch density $A(x - c(\alpha))$ that diminishes with distance $\|x - c(\alpha)\|$ from the arbor center. The postsynaptic activity $a_{TC}(x)$ of target cell $x$ is proportional to the linear sum of its excitation.

$$a_{TC}(x) = \sum_{\alpha} a_{SC}(\alpha) A(x - c(\alpha)) \tag{1}$$

Postsynaptically active target cell bodies release neurotropin, which spreads laterally until consumed by constitutive decay processes. The neurotropin $n(x')$ present at target site $x'$ is assembled from contributions from all active release sites. The contribution of each target cell $x$ is proportional to its postsynaptic activity and weighted by a spreading kernel $N(x - x')$ that is a decreasing function of its distance $\|x - x'\|$ from the measurement site $x'$.

$$n(x') = \sum_{x} a_{TC}(x) N(x - x') \tag{2}$$

A presynaptically active growth cone located at $c(\alpha)$ computes the direction of the local neurotropin gradient by identifying the adjacent lattice node $c'(\alpha) \in \mathcal{C}(c(\alpha))$ with the most neurotropin, where $\mathcal{C}(c(\alpha))$ includes $c(\alpha)$ and its nearest neighbors.

$$c'(\alpha) = \arg\max_{x' \in \mathcal{C}(c(\alpha))} n(x') \tag{3}$$

Once the growth cone has identified $c'(\alpha)$, it swaps positions with the growth cone already located at $c'(\alpha)$, increasing its own neurotropic uptake while preserving a constant synaptic density. Growth cones compute position updates independently, at a rate $\lambda(\alpha) \propto a_{SC}(\alpha) \max_{y \in \mathcal{C}(c(\alpha))} n(x')$. Updates are executed asynchronously, in order of their arrival.

Software simulation of a similar set of equations generates self-organized feature maps when driven by appropriately correlated source cell activity [6]. Here, we illustrate topographic map formation in hardware using correlated spike trains generated by a silicon retina.

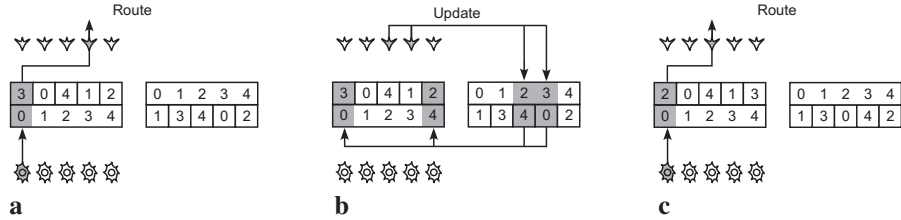

Figure 3: Virtual axon remapping. **a.** Cell bodies tag their spikes with their own source layer addresses, which the forward lookup table translates into target layer destinations. **b.** Axon updates are computed by growth cones, which decode their own target layer addresses through the reverse lookup table to obtain the source layer addresses of their cell bodies that identify their entries in the forward lookup table. **c.** Growth cones move by modifying their entries in the forward and reverse lookup tables to reroute their spikes to updated locations.

# 3 Neurotrope1 system

Our hardware implementation splits the model into three stages: the source layer, the target layer, and the intervening axons (Fig. 2a). Any population of spiking neurons can act as a source layer; in this paper we employ the silicon retina of [7]. The target layer is implemented by a full custom VLSI chip that interleaves a $48 \times 20$ array of growth cone circuits with a $24 \times 20$ array of target cell circuits. There is also a spreading network that represents the intervening medium for propagating neurotropin. The Neurotrope1 chip was fabricated by MOSIS using the TSMC $0.35\mu$m process and has an area of 11.5 mm$^2$. Connections are specified as entries in a pair of lookup tables, stored in an off-chip RAM, that are updated by a Ubicom ip2022 microcontroller as instructed by the Neurotrope1 chip. The ip2022 also controls a USB link that allows a computer to write and read the contents of the RAM. Subsection 3.1 explains how updates are computed by the Neurotrope1 chip and Subsection 3.2 describes the procedure for executing these updates.

## 3.1 Axon updates

Axon updates are computed by the Neurotrope1 chip using the transistor circuits described in [5]. Here, we provide a brief description. The Neurotrope1 chip represents neurotropin as charge spreading through a monolithic transistor channel laid out as a honeycomb lattice. Each growth cone occupies one node of this lattice and extends filopodia to the three adjacent nodes, expressing neurotropin receptors at all four locations (Fig. 2b-c). When a growth cone receives a presynaptic spike, its receptor circuits tap charge from all four nodes onto separate capacitors. The first capacitor voltage to integrate to a threshold resets all of the growth cone's capacitors and transmits a request off-chip to update the growth cone's position by swapping locations with the growth cone currently occupying the winning node.

## 3.2 Address-event remapping

Chips in the Neurotrope1 system exchange spikes encoded in the address-event representation (AER) [8], an asynchronous communication protocol that merges spike trains from every cell on the same chip onto a single shared data link instead of requiring a dedicated wire for each connection. Each spike is tagged with the address of its originating cell for transmission off-chip. Between chips, spikes are routed through a *forward lookup table* that translates their original source layer addresses into their destined target layer addresses

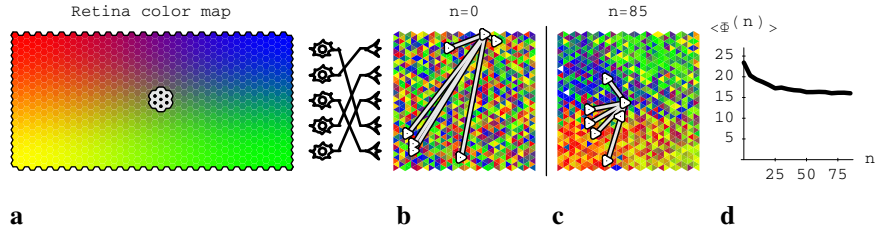

Figure 4: Retinotopic self-organization of ON-center RGCs. **a.** Silicon retina color map of ON-center RGC body positions. A representative RGC body is outlined in white, as are the RGC neighbors that participate in its topographic order parameter $\Phi^{(n)}$. **b.** Target layer color map of growth cone positions for sample $n = 0$, colored by the retinal positions of their cell bodies. Growth cones projected by the representative RGC and its nearest neighbors are outlined in white. Grey lines denote target layer distances used to compute $\Phi^{(n)}$. **c.** Target layer color map at $n = 85$. **d.** Order parameter evolution.

on the receiving chip (Fig. 3a). An axon entry in this forward lookup table is indexed by the source layer address of its cell body and contains the target layer address of its growth cone. The virtual axon moves by updating this entry.

Axon updates are computed by growth cone circuits on the Neurotrope1 chip, encoded as address-events, and sent to the ip2022 for processing. Each update identifies a pair of axon terminals to be swapped. These growth cone addresses are translated through a *reverse lookup table* into the source layer addresses that index the relevant forward lookup table entries (Fig. 3b). Modification of the affected entries in each lookup table completes the axon migration (Fig. 3c).

## 4 Retinotopic self-organization

We programmed the growth cone population to self-organize retinotopic maps by driving them with correlated spike trains generated by the silicon retina. The silicon retina translates patterned illumination in real-time into spike trains that are fed into the Neurotrope1 chip as presynaptic input from different retinal ganglion cell (RGC) types. An ON-center RGC is excited by a spot of light in the center of its receptive field and inhibited by light in the surrounding annulus, while an OFF-center RGC responds analogously to the absence of light. There is an ON-center and an OFF-center RGC located at every retinal coordinate.

To generate appropriately correlated RGC spike trains, we illuminated the silicon retina with various mixtures of light and dark spot stimuli. Each spot stimulus was presented against a uniformly grey background for 100 ms and covered a contiguous cluster of RGCs centered on a pseudorandomly selected position in the retinal plane, eliciting overlapping bursts of spikes whose coactivity established a spatially restricted presynaptic correlation kernel containing enough information to instruct topographic ordering [9]. Strongly driven RGCs could fire at nearly 1 kHz, which was the highest mean rate at which the silicon retina could still be tuned to roughly balance ON- and OFF-center RGC excitability. We tracked the evolution of the growth cone population by reading out the contents of the lookup table every five minutes, a sampling interval selected to include enough patch stimuli to allow each of the $48 \times 20$ possible patches to be activated on average at least once per sample.

We first induced retinotopic self-organization within a single RGC cell type by illuminating the silicon retina with a sequence of randomly centered spots of light presented against a grey background, selectively activating only ON-center RGCs. Each of the 960 growth

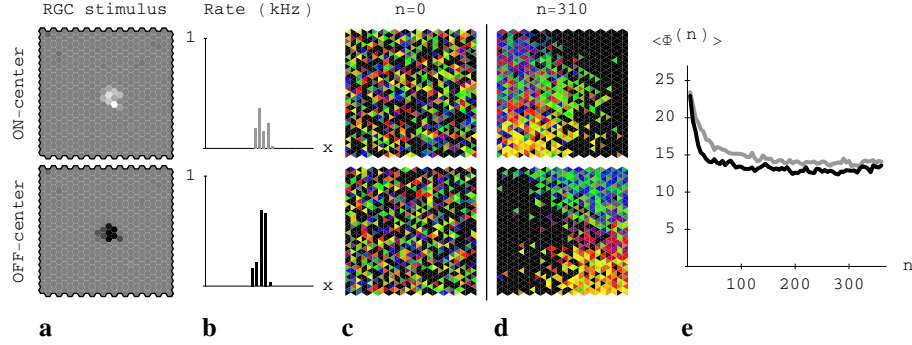

Figure 5: Segregation by cell type under separate light and dark spot stimulation. Top: ON-center; bottom: OFF-center. **a.** Silicon retina image of representative spot stimulus. Light or dark intensity denotes relative ON- or OFF-center RGC output rate. **b.** Spike rates for ON-center (grey) and OFF-center (black) RGCs in column $x$ of a cross-section of a representative spot stimulus. **c.** Target layer color maps of RGC growth cones at sample $n = 0$. Black indicates the absence of a growth cone projected by an RGC of this cell type. Other colors as in Fig. 4. **d.** Target layer color maps at $n = 310$. **e.** Order parameter evolution for ON-center (grey) and OFF-center (black) RGCs.

cones was randomly assigned to a different ON-center RGC, creating a scrambled map from retina to target layer (Fig. 4a-b). The ON-center RGC growth cone population visibly refined the topography of the nonretinotopic initial state (Fig. 4c). We quantify this observation by introducing an *order parameter* $\Phi^{(n)}$ whose value measures the instantaneous retinotopy for an RGC at the $n$th sample. The definition of retinotopy is that adjacent RGCs innervate adjacent target cells, so we define $\Phi^{(n)}$ for a given RGC to be the average target layer distance separating its growth cone from the growth cones projected by the six adjacent RGCs of the same cell type. The population average $\langle \Phi^{(n)} \rangle$ converges to a value that represents the achievable performance on this task (Fig. 4d).

We next induced growth cones projected by each cell type to self-organize disjoint topographic maps by illuminating the silicon retina with a sequence of randomly centered light or dark spots presented against a grey background (Fig. 5a-b). Half the growth cones were assigned to ON-center RGCs and the other half were assigned to the corresponding OFF-center RGCs. We seeded the system with a random projection that evenly distributed growth cones of both cell types across the entire target layer (Fig. 5c). Since only RGCs of the same cell type were coactive, growth cones segregated into ON- and OFF-center clusters on opposite sides of the target layer (Fig. 5d). OFF-center RGCs were slightly more excitable on average than ON-center RGCs, so their growth cones refined their topography more quickly (Fig. 5e) and clustered in the right half of the target layer, which was also more excitable due to poor power distribution on the Neurotrope1 chip.

Finally, we induced growth cones of both cell types to self-organize coordinated retinotopic maps by illuminating the retina with center-surround stimuli that oscillate radially from light to dark or vice versa (Fig. 6). The light-dark oscillation injected enough coactivity between neighboring ON- and OFF-center RGCs to prevent their growth cones from segregating by cell type into disjoint clusters. Instead, both subpopulations developed and maintained coarse retinotopic maps that cover the entire target layer and are oriented in register with one another, properties sufficient to seed more interesting circuits such as oriented receptive fields [10].

Performance in this hardware implementation is limited mainly by variability in the behav-

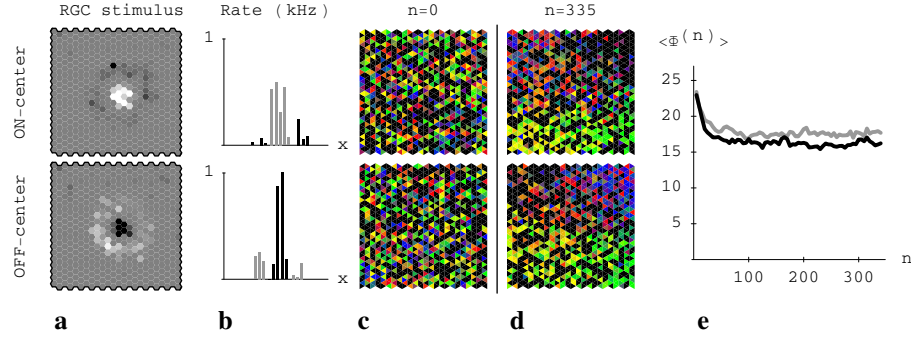

Figure 6: Coordinated retinotopy under center-surround stimulation. Top: ON-center; bottom: OFF-center. **a.** Silicon retina image of a representative center-surround stimulus. Light or dark intensity denotes relative ON- or OFF-center RGC output rate. **b.** Spike rates for ON-center (grey) and OFF-center (black) RGCs in column $x$ of a cross-section of a representative center-surround stimulus. **c.** Target layer color maps of RGC growth cones for sample $n = 0$. Colors as in Fig. 5. **d.** Target layer color maps at $n = 335$. **e.** Order parameter evolution for ON-center (grey) and OFF-center (black) RGCs.

ior of nominally identical circuits on the Neurotrope1 chip and the silicon retina. In the silicon retina, the wide variance of the RGC output rates [7] limits both the convergence speed and the final topographic level achieved by the spot-driven growth cone population. Growth cones move faster when stimulated at higher rates, but elevating the mean output rate of the RGC population allows more excitable RGCs to fire spontaneously at a sustained rate, swamping growth cone-specific guidance signals with stimulus-independent postsynaptic activity that globally attracts all growth cones. The mean RGC output rate must remain low enough to suppress these spontaneous distractors, limiting convergence speed. Variance in the output rates of neighboring RGCs also distorts the shape of the spot stimulus, eroding the fidelity of the correlation-encoded instructions received by the growth cones.

Variability in the Neurotrope1 chip further limits topographic convergence. Migrating growth cones are directed by the local neurotropin landscape, which forms an image of recent presynaptic activity correlations as filtered through the postsynaptic activation of the target cell population. This image is distorted by variations between the properties of individual target cell and neurotropin circuits that are introduced during fabrication. In particular, poor power distribution on the Neurotrope1 chip creates a systematic gradient in target cell excitability that warps a growth cone's impression of the relative coactivity of its neighbors, attracting it preferentially toward the more excitable target cells on the right side of the array.

## 5   Conclusions

In this paper, we demonstrated a completely neuromorphic implementation of retinotopic self-organization. This is the first time every stage of the process has been implemented entirely in hardware, from photon transduction through neural map formation. The only comparable system was described in [11], which processed silicon retina data offline using a software model of neurotrophic guidance running on a workstation. Our system computes results in real time at low power, two prerequisites for autonomous mobile applications.

The novel infrastructure developed to implement virtual axon migration allows silicon growth cones to directly interface with an existing family of AER-compliant devices,

enabling a host of multimodal neuromorphic self-organizing applications. In particular, the silicon retina's ability to translate arbitrary visual stimuli into growth cone-compatible spike trains in real-time opens the door to more ambitious experiments such as using natural video correlations to automatically wire more complicated visual feature maps.

Our faithful adherence to cellular level details yields an algorithm that is well suited to physical implementation. In contrast to all previous self-organizing map chips (e.g. [2, 3]), which implemented a global winner-take-all function to induce competition, our silicon growth cones compute their own updates using purely local information about the neurotropin gradient, a cellular approach that scales effortlessly to larger populations. Performance might be improved by supplementing our purely morphogenetic model with additional physiologically-inspired mechanisms to prune outliers and consolidate well-placed growth cones into permanent synapses.

### Acknowledgments

We would like to thank J. Arthur for developing a USB system to facilitate data collection. This project was funded by the David and Lucille Packard Foundation and the NSF/BITS program (EIA0130822).

## References

[1] T. Kohonen (1982), "Self-organized formation of topologically correct feature maps," *Biol. Cybernetics*, vol. 43, no. 1, pp. 59-69.

[2] W.-C. Fang, B.J. Sheu, O.T.-C. Chen, and J. Choi (1992), "A VLSI neural processor for image data compression using self-organization networks," *IEEE Trans. Neural Networks*, vol. 3, no. 3, pp. 506-518.

[3] S. Rovetta and R. Zunino (1999), "Efficient training of neural gas vector quantizers with analog circuit implementation," *IEEE Trans. Circ. & Sys. II*, vol. 46, no. 6, pp. 688-698.

[4] E.W. Dent and F.B. Gertler (2003), "Cytoskeletal dynamics and transport in growth cone mobility and axon guidance," *Neuron*, vol. 40, pp. 209-227.

[5] B. Taba and K. Boahen (2003), "Topographic map formation by silicon growth cones," in: *Advances in Neural Information Processing Systems 15* (MIT Press, Cambridge, eds. S. Becker, S. Thrun, and K. Obermayer), pp. 1163-1170.

[6] S.Y.M. Lam, B.E. Shi, and K.A. Boahen (2005), "Self-organized cortical map formation by guiding connections," *Proc. 2005 IEEE Int. Symp. Circ. & Sys.*, in press.

[7] K.A. Zaghloul and K. Boahen (2004), "Optic nerve signals in a neuromorphic chip I: Outer and inner retina models," *IEEE Trans. Bio-Med. Eng.*, vol. 51, no. 4, pp. 657-666.

[8] K. Boahen (2000), "Point-to-point connectivity between neuromorphic chips using address-events," *IEEE Trans. Circ. & Sys. II*, vol. 47, pp. 416-434.

[9] K. Miller (1994), "A model for the development of simple cell receptive fields and the ordered arrangement of orientation columns through activity-dependent competition between on- and off-center inputs," *J. Neurosci.*, vol. 14, no. 1, pp. 409-441.

[10] D. Ringach (2004), "Haphazard wiring of simple receptive fields and orientation columns in visual cortex," *J. Neurophys.*, vol. 92, no. 1, pp. 468-476.

[11] T. Elliott and J. Kramer (2002), "Coupling an aVLSI neuromorphic vision chip to a neurotrophic model of synaptic plasticity: the development of topography," *Neural Comp.*, vol. 14, no. 10, pp. 2353-2370.
